# Gaussian and Wishart Hyperkernels

**Risi Kondor, Tony Jebara**
Computer Science Department, Columbia University
1214 Amsterdam Avenue, New York, NY 10027, U.S.A.
{risi,jebara}@cs.columbia.edu

## Abstract

We propose a new method for constructing hyperkenels and define two promising special cases that can be computed in closed form. These we call the Gaussian and Wishart hyperkernels. The former is especially attractive in that it has an interpretable regularization scheme reminiscent of that of the Gaussian RBF kernel. We discuss how kernel learning can be used not just for improving the performance of classification and regression methods, but also as a stand-alone algorithm for dimensionality reduction and relational or metric learning.

## 1 Introduction

The performance of kernel methods, such as Support Vector Machines, Gaussian Processes, etc. depends critically on the choice of kernel. Conceptually, the kernel captures our prior knowledge of the data domain. There is a small number of popular kernels expressible in closed form, such as the Gaussian RBF kernel $k(x, x') = \exp(- \| x - x' \|^2 /(2\sigma^2))$, which boasts attractive and unique properties from an abstract function approximation point of view. In real world problems, however, and especially when the data is heterogenous or discrete, engineering an appropriate kernel is a major part of the modelling process. It is natural to ask whether instead it might be possible to learn the kernel itself from the data.

Recent years have seen the development of several approaches to kernel learning [5][1]. Arguably the most principled method proposed to date is the hyperkernels idea introduced by Ong, Smola and Williamson [8][7][9]. The current paper is a continuation of this work, introducing a new family of hyperkernels with attractive properties.

Most work on kernel learning has focused on finding a kernel which is subsequently to be used in a conventional kernel machine, turning learning into an essentially two-stage process: first learn the kernel, then use it in a conventional algorithm such as an SVM to solve a classification or regression task. Recently there has been increasing interest in using the kernel in its own right to answer *relational* questions about the dataset. Instead of predicting individual labels, a kernel characterizes which pairs of labels are likely to be the same, or related. Kernel learning can be used to infer the network structure underlying data. A different application is to use the learnt kernel to produce a low dimensional embedding via kernel PCA. In this sense, kernel learning can be also be regarded as a dimensionality reduction or metric learning algorithm.

## 2 Hyperkernels

We begin with a brief review of the kernel and hyperkernel formalism. Let $\mathcal{X}$ be the input space, $\mathcal{Y}$ the output space, and $\{(x_1, y_1), (x_2, y_2), \ldots, (x_m, y_m)\}$ the training data. By kernel we mean a symmetric function $k \colon \mathcal{X} \times \mathcal{X} \to \mathbb{R}$ that is positive definite on $\mathcal{X}$. Whenever

we refer to a function being positive definite, we assume that it is also symmetric. Positive definiteness guarantees that $k$ induces a Reproducing Kernel Hilbert Space (RKHS) $\mathcal{F}$, which is a vector space of functions spanned by $\{\, k_x(\cdot) = k(x, \cdot) \mid x \in \mathcal{X} \,\}$ and endowed with an inner product satisfying $\langle k_x, k_{x'} \rangle = k(x, x')$. Kernel-based learning algorithms find a hypothesis $\hat{f} \in \mathcal{F}$ by solving some variant of the Regularized Risk Minimzation problem

$$\hat{f} = \arg\min_{f \in \mathcal{F}} \left[ \frac{1}{m} \sum_{i=1}^{m} L(f(x_i), y_i) + \frac{1}{2} \| f \|_{\mathcal{F}}^2 \right]$$

where $L$ is a loss function of our choice. By the Representer Theorem [2], $\hat{f}$ is expressible in the form $\hat{}(x) = \sum_{i=1}^{m} \alpha_i\, k(x_i, x)$ for some $\alpha_1, \alpha_2, \ldots, \alpha_m \in \mathbb{R}$.

The idea expounded in [8] is to set up an analogous optimization problem for finding $k$ itself in the RKHS of a hyperkernel $K \colon \underline{\mathcal{X}} \times \underline{\mathcal{X}} \to \mathbb{R}$, where $\underline{\mathcal{X}} = \mathcal{X}^2$. We will sometimes view $K$ as a function of four arguments, $K((x_1, x_1'), (x_2, x_2'))$, and sometimes as a function of two pairs, $K(\underline{x}_1, \underline{x}_2)$, with $\underline{x}_1 = (x_1, x_1')$ and $\underline{x}_2 = (x_2, x_2')$. To induce an RKHS $K$ must be positive definite in the latter sense. Additionaly, we have to ensure that the solution of our regularized risk minimization problem is itself a kernel. To this end, we require that the functions $K_{x_1, x_1'}(x_2, x_2')$ that we get by fixing the first two arguments of $K((x_1, x_1'), (x_2, x_2'))$ be symmetric and positive definite kernel in the remaining two arguments.

**Definition 1.** *Let $\mathcal{X}$ be a nonempty set, $\underline{\mathcal{X}} = \mathcal{X} \times \mathcal{X}$ and $K \colon \underline{\mathcal{X}} \times \underline{\mathcal{X}} \to \mathbb{R}$ with $K_{\underline{x}}(\,\cdot\,) = K(\underline{x}, \cdot) = K(\cdot, \underline{x})$. Then $K$ is called a **hyperkernel** on $\mathcal{X}$ if and only if*

1. *$K$ is positive definite on $\underline{\mathcal{X}}$ and*

2. *for any $\underline{x} \in \underline{\mathcal{X}}$, $K_{\underline{x}}$ is positive definite on $\mathcal{X}$.*

Denoting the RKHS of $K$ by $\mathcal{K}$, potential kernels lie in the cone $\mathcal{K}^{pd} = \{\, k \in \mathcal{K} \mid k \text{ is pos.def.} \,\}$. Unfortunately, there is no simple way of restricting kernel learning algorithms to $\mathcal{K}^{pd}$. Instead, we will restrict ourselves to the positive quadrant $\mathcal{K}^+ = \{\, k \in \mathcal{K} \mid \langle k, K_{\underline{x}} \rangle \geq 0 \;\; \forall\, \underline{x} \in \underline{\mathcal{X}} \,\}$, which is a subcone of $\mathcal{K}^{pd}$.

The actual learning procedure involved in finding $k$ is very similar to conventional kernel methods, except that now regularized risk minimization is to be performed over all *pairs* of data points:

$$\hat{k} = \arg\min_{\mathcal{K}^*} \left[ Q(X, Y, k) + \frac{1}{2} \| k \|_{\mathcal{K}}^2 \right], \tag{1}$$

where $Q$ is a quality functional describing how well $k$ fits the training data and $\mathcal{K}^* = \mathcal{K}^+$. Several candidates for $Q$ are described in [8].

If $\mathcal{K}^*$ has the property that for any $S \subset \underline{\mathcal{X}}$ the orthogonal projection of any $k \in \mathcal{K}^*$ to the subspace spanned by $\{\, K_{\underline{x}} \mid \underline{x} \in \underline{\mathcal{X}} \,\}$ remains in $\mathcal{K}^*$, then $\widehat{k}$ is expressible as

$$\widehat{k}(x, x') = \sum_{i,j=1}^{m} \alpha_{ij}\, K_{(x_i, x_j)}(x, x') = \sum_{i,j=1}^{m} \alpha_{ij}\, K((x_i, x_j), (x, x')) \tag{2}$$

for some real coefficients $(\alpha_{ij})_{i,j}$. In other words, we have a hyper-representer theorem. It is easy to see that for $\mathcal{K}^* = \mathcal{K}^+$ this condition is satisfied provided that $K((x_1, x_1'), (x_2, x_2')) \geq 0$ for all $x_1, x_1', x_2, x_2' \in \mathcal{X}$. Thus, in this case to solve (1) it is sufficient to optimize the variables $(\alpha_{ij})_{i,j=1}^{m}$, introducing the additional constraints $\alpha_{ij} \geq 0$ to enforce $\widehat{k} \in \mathcal{K}^+$.

Finding functions that satisfy Definition 1 and also make sense in terms of regularization theory or practical problem domains in not trivial. Some potential choices are presented in [8]. In this paper we propose some new families of hyperkernels. The key tool we use is the following simple lemma.

**Lemma 1.** *Let $\{g_z : \mathcal{X} \to \mathbb{R}\}$ be a family of functions indexed by $z \in \mathcal{Z}$ and let $h \colon \mathcal{Z} \times \mathcal{Z} \to \mathbb{R}$ be a kernel. Then*

$$k(x, x') = \int \int g_z(x)\, h(z, z')\, g_{z'}(x')\, dz\, dz' \tag{3}$$

is a kernel on $\mathcal{X}$. Furthermore, if $h$ is pointwise positive $(h(z, z') \geq 0)$ and $\{\, g_z \colon \mathcal{X} \times \mathcal{X} \to \mathbb{R} \,\}$ is a family of pointwise positive kernels, then

$$K\left((x_1, x_1'), (x_2, x_2')\right) = \int \int g_{z_1}(x_1, x_1') \, h(z_1, z_2) \, g_{z_2}(x_2, x_2') \, dz_1 \, dz_2 \qquad (4)$$

is a hyperkernel on $\mathcal{X}$, and it satisfies $K((x_1, x_1'), (x_2, x_2')) \geq 0$ for all $x_1, x_1', x_2, x_2' \in \mathcal{X}$.

## 3  Convolution hyperkernels

One interpretation of a kernel $k(x, x')$ is that it quantifies some notion of similarity between points $x$ and $x'$. For the Gaussian RBF kernel, and heat kernels in general, this similarity can be regarded as induced by a diffusion process in the ambient space [4]. Just as physical substances diffuse in space, the similarity between $x$ and $x'$ is mediated by intermediate points, in the sense that by virtue of $x$ being similar to some $x_0$ and $x_0$ being similar to $x'$, $x$ and $x'$ themselves become similar to each other. This captures the natural transitivity of similarity. Specifically, the normalized Gaussian kernel on $\mathbb{R}^n$ of variance $2t = \sigma^2$,

$$k_t(x, x') = \frac{1}{(4\pi t)^{n/2}} \, e^{-\|x - x'\|^2 / (4t)},$$

satisfies the well known convolution property

$$k_t(x, x') = \int k_{t/2}(x, x_0) \, k_{t/2}(x_0, x) \, dx_0 \,. \qquad (5)$$

Such kernels are by definition homogenous and isotropic in the ambient space.

What we hope for from the hyperkernels formalism is to be able to adapt to the inhomogeneous and anisotropic nature of training data, while retaining the transitivity idea in some form. Hyperkernels achieve this by weighting the integrand of (5) in relation to what is "on the other side" of the hyperkernel. Specifically, we define convolution hyperkernels by setting

$$g_z(x, x') = r(x, z) \, r(x', z)$$

in (4) for some $r \colon \mathcal{X} \times \mathcal{X} \to \mathbb{R}$. By (3), the resulting hyperkernel always satisfies the conditions of Definition 1.

**Definition 2.** *Given functions* $r \colon \mathcal{X} \times \mathcal{X} \to \mathbb{R}$ *and* $h \colon \mathcal{X} \times \mathcal{X} \to \mathbb{R}$ *where* $h$ *is positive definite, the* ***convolution hyperkernel*** *induced by* $r$ *and* $h$ *is*

$$K\left((x_1, x_1'), (x_2, x_2')\right) = \int \int r(x_1, z_1) \, r(x_1', z_1) \, h(z_1, z_2) \, r(x_2, z_2) \, r(x_2', z_2) \, dz_1 \, dz_2 \,. \qquad (6)$$

A good way to visualize the structure of convolution hyperkernels is to note that (6) is proportional to the likelihood of the graphical model in the figure to the right. The only requirements on the graphical model are to have the same potential function $\psi_1$ at each of the extremities and to have a positive definite potential function $\psi_2$ at the core.

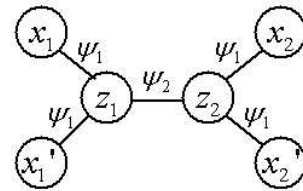

### 3.1  The Gaussian hyperkernel

To make the foregoing more concrete we now investigate the case where $r(x, x')$ and $h(z, z')$ are Gaussians. To simplify the notation we use the shorthand

$$\langle x, x' \rangle_{\sigma^2} = \frac{1}{(2\pi\sigma^2)^{n/2}} \, e^{-\|x - x'\|^2 / (2\sigma^2)}.$$

The **Gaussian hyperkernel** on $\mathcal{X} = \mathbb{R}^n$ is then defined as

$$K((x_1, x_1'), (x_2, x_2')) = \int_{\mathcal{X}} \int_{\mathcal{X}} \langle x_1, z \rangle_{\sigma^2} \, \langle z, x_1' \rangle_{\sigma^2} \, \langle z, z' \rangle_{\sigma_h^2} \, \langle x_2, z' \rangle_{\sigma^2} \, \langle z', x_2' \rangle_{\sigma^2} \, dz \, dz'. \qquad (7)$$

Fixing $x$ and completing the square we have

$$\langle x_1, z \rangle_{\sigma^2} \langle z, x_1' \rangle_{\sigma^2} = \frac{1}{(2\pi\sigma^2)^n} \exp\left( -\frac{1}{2\sigma^2}\left( \|z-x_1\|^2 + \|z-x_1'\|^2 \right) \right) =$$

$$\frac{1}{(2\pi\sigma^2)^n} \exp\left( -\frac{1}{\sigma^2} \left\| z - \frac{x_1+x_1'}{2} \right\|^2 - \frac{\|x_1-x_1'\|^2}{4\sigma^2} \right) = \langle x_1, x_1' \rangle_{2\sigma^2} \langle z, \overline{x}_1 \rangle_{\sigma^2/2} ,$$

where $\overline{x}_i = (x_i + x_i')/2$. By the convolution property of Gaussians it follows that

$$K((x_1, x_1'), (x_2, x_2')) =$$

$$\langle x_1, x_1' \rangle_{2\sigma^2} \langle x_2, x_2' \rangle_{2\sigma^2} \int_{\mathcal{X}} \int_{\mathcal{X}} \langle \overline{x}_1, z \rangle_{\sigma^2/2} \langle z, z' \rangle_{\sigma_h^2} \langle z, \overline{x}_2 \rangle_{\sigma^2/2} \, dz \, dz' =$$

$$\langle x_1, x_1' \rangle_{2\sigma^2} \langle x_2, x_2' \rangle_{2\sigma^2} \langle \overline{x}_1, \overline{x}_2 \rangle_{\sigma^2+\sigma_h^2} . \quad (8)$$

It is an important property of the Gaussian hyperkernel that it can be evaluated in closed form. A noteworthy special case is when $h(x, x') = \delta(x, x')$, corresponding to $\sigma_h^2 \to 0$. At the opposite extreme, in the limit $\sigma_h^2 \to \infty$, the hyperkernel decouples into the product of two RBF kernels.

Since the hyperkernel expansion (2) is a sum over hyperkernel evaluations with one pair of arguments fixed, it is worth examining what these functions look like:

$$K_{x_1, x_1'}(x_2, x_2') \; \propto \; \exp\left( -\frac{\|\overline{x}_1 - \overline{x}_2\|^2}{2(\sigma^2 + \sigma_h^2)} \right) \exp\left( -\frac{\|x_2 - x_2'\|^2}{2\sigma'^2} \right) \quad (9)$$

with $\sigma' = \sqrt{2}\sigma$. This is really a conventional Gaussian kernel between $x_2$ and $x_2'$ multiplied by a spatially varying Gaussian intensity factor depending on how close the *mean* of $x_2$ and $x_2'$ is to the mean of the training pair. This can be regarded as a **localized Gaussian**, and the full kernel (2) will be a sum of such terms with positive weights. As $x_2$ and $x_2'$ move around in $\mathcal{X}$, whichever localized Gaussians are centered close to their mean will dominate the sum. By changing the $(\alpha_{ij})$ weights, the kernel learning algorithm can choose $k$ from a highly flexible class of potential kernels.

The close relationship of $K$ to the ordinary Gaussian RBF kernel is further borne out by changing coordinates to $\hat{x} = (x + x')/\sqrt{2}$ and $\tilde{x} = (x - x')/\sqrt{2}$, which factorizes the hyperkernel in the form

$$K((\hat{x}_1, \tilde{x}_1), (\hat{x}_2, \tilde{x}_2)) = \hat{K}(\hat{x}_1, \hat{x}_2)\tilde{K}(\tilde{x}_1, \tilde{x}_2) = \left[ \langle \hat{x}_1, \hat{x}_2 \rangle_{2(\sigma^2+\sigma_h^2)} \right] \left[ \langle \tilde{x}_1, 0 \rangle_{\sigma^2} \langle \tilde{x}_2, 0 \rangle_{\sigma^2} \right].$$

Omitting details for brevity, the consequences of this include that $\mathcal{K} = \hat{\mathcal{K}} \times \tilde{\mathcal{K}}$, where $\hat{\mathcal{K}}$ is the RKHS of a Gaussian kernel over $\mathcal{X}$, while $\tilde{\mathcal{K}}$ is the one-dimensional space generated by $\langle \tilde{x}, 0 \rangle_{\sigma^2}$: each $k \in \mathcal{K}$ can be written as $k(\hat{x}, \tilde{x}) = \hat{k}(\hat{x}) \langle \tilde{x}, 0 \rangle_{\sigma^2}$. Furthermore, the regularization operator $\Upsilon$ (defined by $\langle k, k' \rangle_{\mathcal{K}} = \langle \Upsilon k, \Upsilon k' \rangle_{L_2}$ [10]) will be

$$\langle \tilde{x}, 0 \rangle_{\sigma^2} \int \widehat{\kappa}(\omega) \, e^{i\omega x} d\omega \quad \mapsto \quad \langle \tilde{x}, 0 \rangle_{\sigma^2} \int e^{(\sigma^2+\sigma_h^2)\,\omega^2/2} \, \widehat{\kappa}(\omega) \, e^{i\omega x} d\omega$$

where $\widehat{\kappa}(\omega)$ is the Fourier transform of $\widehat{k}(\widehat{x})$, establishing the same exponential regularization penalty scheme in the Fourier components of $\hat{k}$ that is familiar from the theory of Gaussian RBF kernels. In summary, $K$ behaves in $(\hat{x}_1, \hat{x}_2)$ like a Gaussian kernel with variance $2(\sigma^2 + \sigma_h^2)$, but in $\tilde{x}$ it just effects a one-dimensional feature mapping.

## 4   Anisotropic hyperkernels

With the hyperkernels so far far we can only learn kernels that are a sum of rotationally invariant terms. Consequently, the learnt kernel will have a locally isotropic character. Yet, rescaling of the axes and anisotropic dilations are one of the most common forms of variation in naturally occurring data that we would hope to accomodate by learning the kernel.

### 4.1 The Wishart hyperkernel

We define the **Wishart hyperkernel** as

$$K((x_1, x_1'), (x_2, x_2')) = \int_{\Sigma \succeq 0} \int_{\mathcal{X}} \langle x_1, z \rangle_\Sigma \, \langle z, x_1' \rangle_\Sigma \, \langle x_2, z \rangle_\Sigma \, \langle z, x_2' \rangle_\Sigma \, \mathcal{IW}(\Sigma; C, r) \, dz \, d\Sigma. \quad (10)$$

where

$$\langle x, x' \rangle_\Sigma = \frac{1}{(2\pi)^{n/2} \, |\Sigma|^{1/2}} \, e^{-(x-x')^\top \Sigma^{-1} (x-x')/2},$$

and $\mathcal{IW}(\Sigma; C, r)$ is the inverse Wishart distribution

$$\frac{|C|^{r/2}}{Z_{r,n} \, |\Sigma|^{(n+r+1)/2}} \, \exp\left(-\mathrm{tr}\left(\Sigma^{-1} C\right)/2\right)$$

over positive definite matrices (denoted $\Sigma \succeq 0$) [6]. Here $r$ is an integer parameter, $C$ is an $n \times n$ positive definite parameter matrix and $Z_{r,n} = 2^{rn/2} \pi^{n(n-1)/4} \prod_{i=1}^{n} \Gamma((r+1-i)/2)$ is a normalizing factor. The Wishart hyperkernel can be seen as the anisotropic analog of (7) in the limit $\sigma_h^2 \to 0$, $\langle z, z' \rangle_{\sigma_h^2} \to \delta(z, z')$. Hence, by Lemma 1, it is a valid hyperkernel. In analogy with (8),

$$K((x_1, x_1'), (x_2, x_2')) = \int_{\Sigma \succeq 0} \langle x_1, x_1' \rangle_{2\Sigma} \, \langle x_2, x_2' \rangle_{2\Sigma} \, \langle \overline{x}_1, \overline{x}_2 \rangle_\Sigma \, \mathcal{IW}(\Sigma; C, r) \, d\Sigma. \quad (11)$$

By using the identity $v^\top A \, v = \mathrm{tr}(A(vv^\top))$,

$$\langle x, x' \rangle_\Sigma \, \mathcal{IW}(\Sigma; C, r) = \frac{|C|^{r/2}}{(2\pi)^{n/2} Z_{r,n} \, |\Sigma|^{(n+r+2)/2}} \, \exp\left(-\mathrm{tr}\left(\Sigma^{-1}(C+S)\right)/2\right) =$$

$$\frac{Z_{r+1,n}}{(2\pi)^{n/2} Z_{r,n}} \frac{|C|^{r/2}}{|C+S|^{(r+1)/2}} \, \mathcal{IW}(\Sigma; C+S, r+1),$$

where $S = (x-x')(x-x')^\top$. Cascading this through each of the terms in the integrand of (11) and noting that the integral of a Wishart density is unity, we conclude that

$$K((x_1, x_1'), (x_2, x_2')) \propto \frac{|C|^{r/2}}{|C+S_{\mathrm{tot}}|^{(r+3)/2}}, \quad (12)$$

where $S_{\mathrm{tot}} = S_1 + S_2 + S_*$; $S_i = \frac{1}{2}(x_i - x_i')(x_i - x_i')^\top$; and $S_* = (\overline{x}_1 - \overline{x}_2)(\overline{x}_1 - \overline{x}_2)^\top$. We can read off that for given $\|x_1 - x_1'\|$, $\|x_2 - x_2'\|$, and $\|\overline{x} - \overline{x}'\|$, the hyperkernel will favor quadruples where $x_1 - x_1'$, $x_2 - x_2'$, and $\overline{x} - \overline{x}'$ are close to parallel to each other and to the largest eigenvector of $C$. It is not so easy to immediately see the dependence of $K$ on the relative distances between $x_1, x_1', x_2$ and $x_2'$.

To better expose the qualitative behavior of the Wishart hyperkernel, we fix $(x_1, x_1')$, assume that $C = cI$ for some $c \in \mathbb{R}$ and use the identity $\left|cI + vv^\top\right| = c^{n-1}\left(c + \|v\|^2\right)$ to write

$$K_{x_1, x_1'}(x_2, x_2') \propto \left[\frac{Q_c(2S_1, 2S_*)}{\left(c + 4\|\overline{x}_1 - \overline{x}_2\|^2\right)^{1/4}}\right]^{(r+3)/2} \left[\frac{Q_c(S_1 + S_*, S_2)}{\left(c + \|x_2 - x_2'\|^2\right)^{1/4}}\right]^{r+3}$$

where $Q_c(A, B)$ is the affinity

$$Q_c(A, B) = \frac{|cI + 2A|^{1/4} \cdot |cI + 2B|^{1/4}}{|cI + A + B|^{1/2}}.$$

This latter expression is a natural positive definite similarity metric between positive definite matrices, as we can see from the fact that it is the overlap integral (Bhattacharyya kernel)

$$Q_c(A, B) = \int \left[\langle x, 0 \rangle_{(cI+2A)^{-1}}\right]^{1/2} \left[\langle x, 0 \rangle_{(cI+2B)^{-1}}\right]^{1/2} dx$$

between two zero-centered Gaussian distributions with inverse covariances $cI+2A$ and $cI+2B$, respectively [3].

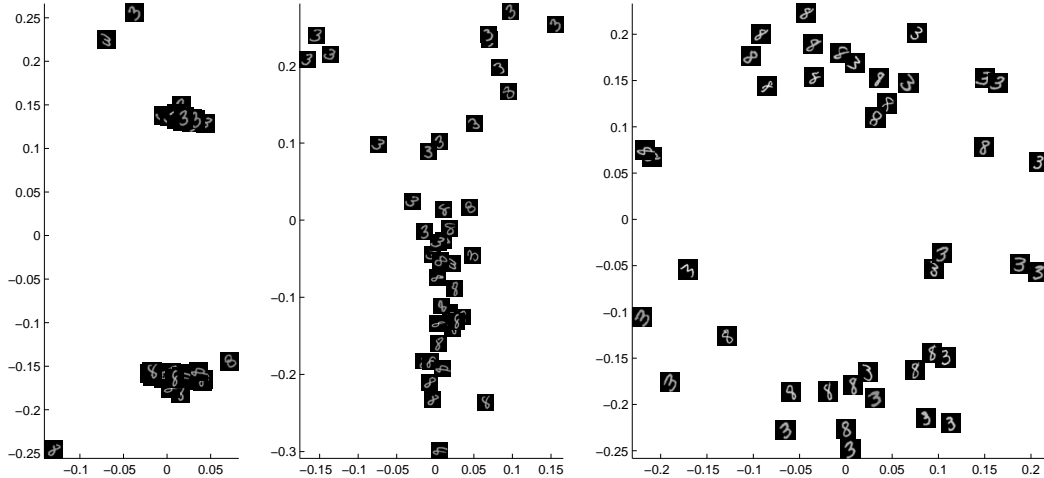

Figure 1: The first two panes show the separation of '3's and '8's in the training and testing sets respectively achieved by the Gaussian hyperkernel (the plots show the data plotted by its first two eigenvectors according to the learned kernel $k$). The right hand pane shows a similar KernelPCA plot but based on a fixed RBF kernel.

## 5  Experiments

We conducted preliminary experiments with the hyperkernels in relation learning between pairs of datapoints. The idea here is that the learned kernel $k$ naturally induces a distance metric $d(x,x') = \sqrt{k(x,x) - 2k(x,x') + k(x',x')}$, and in this sense kernel learning is equivalent to learning $d$. Given a labeled dataset, we can learn a kernel which effectively remaps the data in such a way that data points with the same label are close to each other, while those with different labels are far apart.

For classification problems ($y_i$ being the class label), a natural choice of quality functional similar to the hinge loss is $Q(X,Y,k) = \frac{1}{m^2}\sum_{i,j=1}^{m} |1 - y_{ij}k(x_i,x_j)|_+$, where $|z|_+ = z$ if $z \geq 0$ and $|z|_+ = 0$ for $z < 0$, while $y_{ij} = 1$ if $y_i = y_j$. The corresponding optimization problem learns $k(x,x') = \sum_{i=1}^{m}\sum_{j=1}^{m} \alpha_{ij}K((x,x'),(x_i,x_j)) + b$ minimizing

$$\frac{1}{2}\sum_{i,j}\sum_{i',j'} \alpha_{ij}\alpha_{i'j'} K((x_i,x_j),(x_{i'},x_{j'})) + C\sum_{i,j}\xi_{ij}$$

subject to the classification constraints

$$y_{ij}\Big(\sum_{i',j'}\alpha_{i'j'} K((x_{i'},x_{j'}),(x_i,x_j)) + b\Big) \geq 1 - \xi_{ij} \qquad \xi_{ij} \geq 0 \qquad \alpha_{ij} \geq 0$$

for all pairs of $i,j \in \{1,2,\ldots,m\}$. In testing we interpret $k(x,x') > 0$ to mean that $x$ and $x'$ are of the same class and $k(x,x') \leq 0$ to mean that they are of different classes.

As an illustrative example we learned a kernel (and hence, a metric) between a subset of the NIST handwritten digits[1]. The training data consisted of 20 '3's and 20 '8's randomly rotated by $\pm 45$ degrees to make the problem slightly harder. Figure 1 shows that a kernel learned by the above strategy with a Gaussian hyperkernel with parameters set by cross validation is extremely good at separating the two classes in training as well as testing. In comparison, in a similar plot for a fixed RBF kernel the '3's and '8's are totally intermixed. Interpreting this as an information retrieval problem, we can imagine inflating a ball around each data point in the test set and asking how many other data points in this ball are of the same class. The corresponding area under the curve (AUC) in the original space is just 0.5575, while in the hyperkernel space it is 0.7341.

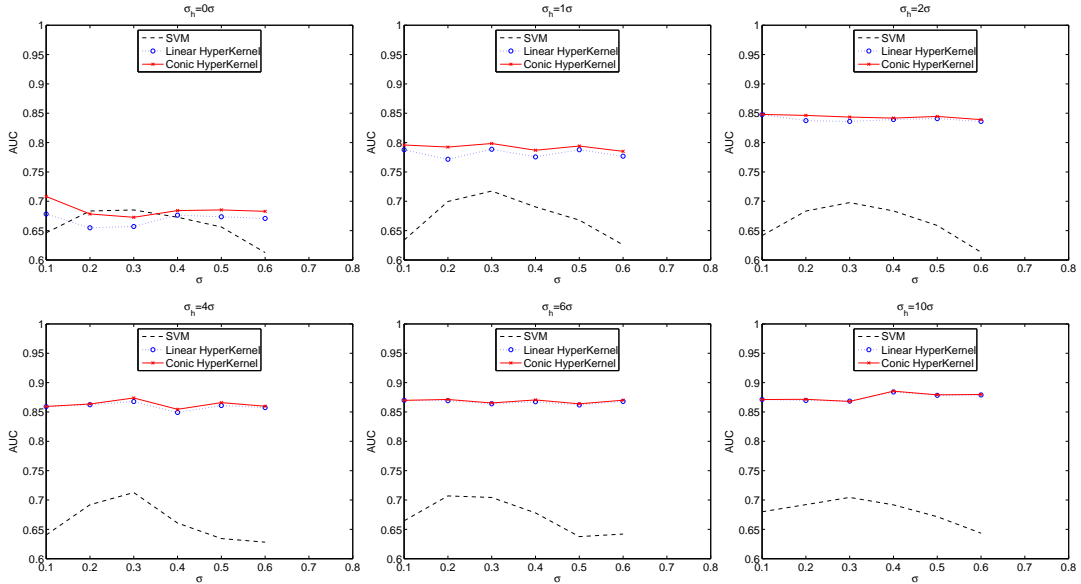

Figure 2: Test area under the curve (AUC) for Olivetti face recognition under varying $\sigma$ and $\sigma_h$.

We ran a similar experiment but with multiple classes on the Olivetti faces dataset, which consists of $92 \times 112$ pixel normalized gray-scale images of 30 individuals in 10 different poses. Here we also experimented with dropping the $\alpha_{ij} \geq 0$ constraints, which breaks the positive definiteness of $k$, but might still give a reasonable similarity measure. The first case we call "conic hyperkernels", whereas the second are just "linear hyperkernels". Both involve solving a quadratic program over $2m^2 + 1$ variables. Finally, as a baseline, we trained an SVM over *pairs* of datapoints to predict $y_{ij}$, representing $(x_i, x_j)$ with a concatenated feature vector $[x_i, x_j]$ and using a Gaussian RBF between these concatenations.

The results on the Olivetti dataset are summarized in Figure 2. We trained the system with $m = 20$ faces and considered all pairs of the training data-points (i.e. 400 constraints) to find a kernel that predicted the labeling matrix. When speed becomes an issue it often suffices to work with a subsample of the binary entries in the $m \times m$ label matrix and thus avoid having $m^2$ constraints. Also, we only need to consider half the entries due to symmetry. Using the learned kernel, we then test on 100 unseen faces and predict all their pairwise kernel evaluations, in other words, $10^4$ predicted pair-wise labelings. Test error rates are averaged over 10 folds of the data. For both the baseline Gaussian RBF and the Gaussian hyperkernels we varied the $\sigma$ parameter from 0.1 to 0.6. For the Gaussian hyperkernel we also varied $\sigma_h$ from 0 to $10\sigma$. We used a value of $C = 10$ for all experiments and for all algorithms. The value of $C$ had very little effect on the testing accuracy.

Using a conic hyperkernel combination did best in labeling new faces. The advantage over SVMs is dramatic. The support vector machine can only achieve an AUC of less than 0.75 while the Gaussian hyperkernel methods achieve an AUC of almost 0.9 with only $T = 20$ training examples. While the difference between the conic and linear hyperkernel methods is harder to see, across all settings of $\sigma$ and $\sigma_h$, the conic combination outperformed the linear combination over 92% of the time. The conic hyperkernel combination is also the only method of the three that guarantees a true Mercer kernel as an output which can then be converted into a valid metric. The average runtime for the three methods was comparable. The SVM took $2.08s \pm 0.18s$, the linear hyperkernel took $2.75s \pm 0.10s$ and the conic hyperkernel took $7.63s \pm 0.50s$ to train on $m = 20$ faces with $m^2$ constraints. We implemented quadratic programming using the MOSEK optimization package on a single CPU workstation.

# 6    Conclusions

The main barrier to hyperkernels becoming more popular is their high computational demands (out of the box algorithms run in $O(m^6)$ time as opposed to $O(m^3)$ in regular learning). In certain metric learning and on-line settings however this need not be forbidding, and is compensated for by the elegance and generality of the framework.

The Gaussian and Wishart hyperkernels presented in this paper are in a sense canonical, with intuitively appealing interpretations. In the case of the Gaussian hyperkernel we even have a natural regularization scheme. Preliminary experiments show that these new hyperkernels can capture the inherent structure of some input spaces. We hope that their introduction will give a boost to the whole hyperkernels field.

## Acknowledgements

The authors wish to thank Zoubin Ghahramani, Alex Smola and Cheng Soon Ong for discussions related to this work. This work was supported in part by National Science Foundation grants IIS-0347499, CCR-0312690 and IIS-0093302.

## Footnotes

[1]Provided at http://yann.lecun.com/exdb/mnist/ courtesy of Yann LeCun and Corinna Cortes.

## References

[1] N. Cristianini, J. Shawe-Taylor, A. Elisseeff, and J. Kandola. On kernel-target alignment. In T. G. Dietterich, S. Becker, and Z. Ghahramani, editors, *Advances in Neural Information Processing Systems 14*, pages 367 – 373, Cambridge, MA, 2002. MIT Press.

[2] G. S. Kimeldorf and G. Wahba. Some results on Tchebycheffian spline functions. *J. Math. Anal. Applic.*, 33:82–95, 1971.

[3] R. Kondor and T. Jebara. A kernel between sets of vectors. In *Machine Learning: Tenth International Conference, ICML 2003*, 2003.

[4] R. Kondor and J. Lafferty. Diffusion kernels on graphs and other discrete input spaces. In *Machine Learning: Proceedings of the Nineteenth International Conference (ICML '02)*, 2002.

[5] G. Lanckriet, N. Cristianini, P. Bartlett, L. El Ghaoui, and M. I. Jordan. Learning the kernel matrix with semi-definite programming. *Journal of Machine Learning Research*, 5:27 – 72, 2004.

[6] T. P. Minka. Inferring a Gaussian distribution, 2001. Tutorial paper available at `http://www.stat.cmu.edu/ minka/papers/learning.html`.

[7] C. S. Ong and A. J. Smola. Machine learning using hyperkernels. In *Proceedings of the International Conference on Machine Learning*, 2003.

[8] Cheng Soon Ong, Alexander J. Smola, and Robert C. Williamson. Hyperkernels. In S. Thrun S. Becker and K. Obermayer, editors, *Advances in Neural Information Processing Systems 15*, pages 478–485. MIT Press, Cambridge, MA, 2003.

[9] Cheng Soon Ong, Alexander J. Smola, and Robert C. Williamson. Learning the kernel with hyperkernels. Sumbitted to the Journal of Machine Learning Research, 2003.

[10] B. Schölkopf and A. J. Smola. *Learning with Kernels*. MIT Press, 2002.
